# Selective Integration: A Model for Disparity Estimation

**Michael S. Gray, Alexandre Pouget, Richard S. Zemel,**
**Steven J. Nowlan, Terrence J. Sejnowski**
Departments of Biology and Cognitive Science
University of California, San Diego
La Jolla, CA   92093
and
Howard Hughes Medical Institute
Computational Neurobiology Lab
The Salk Institute, P. O. Box 85800
San Diego, CA   92186-5800
Email: michael, alex, zemel, nowlan, terry@salk.edu

## Abstract

Local disparity information is often sparse and noisy, which creates two conflicting demands when estimating disparity in an image region: the need to spatially average to get an accurate estimate, and the problem of not averaging over discontinuities. We have developed a network model of disparity estimation based on disparity-selective neurons, such as those found in the early stages of processing in visual cortex. The model can accurately estimate multiple disparities in a region, which may be caused by transparency or occlusion, in real images and random-dot stereograms. The use of a selection mechanism to selectively integrate reliable local disparity estimates results in superior performance compared to standard back-propagation and cross-correlation approaches. In addition, the representations learned with this selection mechanism are consistent with recent neurophysiological results of von der Heydt, Zhou, Friedman, and Poggio [8] for cells in cortical visual area V2. Combining multi-scale biologically-plausible image processing with the power of the mixture-of-experts learning algorithm represents a promising approach that yields both high performance and new insights into visual system function.

## 1   INTRODUCTION

In many stereo algorithms, the local correlation between images from the two eyes is used to estimate relative depth (Jain, Kasturi, & Schunk [5]). Local correlation measures, however, convey no information about the reliability of a particular disparity measurement. In the model presented here, we introduce a separate *selection* mechanism to determine which locations of the visual input have consistent disparity information. The focus was on several challenging viewing situations in which disparity estimation is not straightforward. For example, can the model estimate the disparity of more than one object in a scene? Does occlusion lead to poorer disparity estimation? Can the model determine the disparities of two transparent surfaces? Does the model estimate accurately the disparities present in real world images? Datasets corresponding to these different conditions were generated and used to test the model.

Our goal is to develop a neurobiologically plausible model of stereopsis that accurately estimates disparity. Compared to traditional cross-correlation approaches that try to compute a depth map for all locations in space, the mixture-of-experts model used here searches for sparse, reliable patterns or configurations of disparity stimuli that provide evidence for objects at different depths. This allows partial segmentation of the image to obtain a more compact representation of disparities. Local disparity estimates are sufficient in this case, as long as we selectively segment those regions of the image with reliable disparity information.

The rest of the paper is organized as follows. First, we describe the architecture of the mixture-of-experts model. Second, we provide a brief qualitative description of the model's performance followed by quantitative results on a variety of datasets. In the third section, we compare the activity of units in the model to recent neurophysiological data. Finally, we discuss these findings, and consider remaining open questions.

## 2   MIXTURE-OF-EXPERTS MODEL

The model of stereopsis that we have explored is based on the filter model for motion detection devised by Nowlan and Sejnowski [6]. The motion problem was readily adapted to stereopsis by changing the time domain of motion to the left/right image domain for stereopsis. Our model (Figure 1) consisted of several stages and computed its output using only feed-forward processing, as described below (see also Gray, Pouget, Zemel, Nowlan, and Sejnowski [2] for more detail). The output of the first stage (disparity energy filters) became the input to two different primary pathways: (1) the local disparity networks, and (2) the selection networks. The activation of each of the four disparity-tuned output units in the model was the product of the outputs of the two primary pathways (summed across space). An objective function based on the mixture-of-experts framework (Jacobs, Jordan, Nowlan, & Hinton [4]) was used to optimize the weights from the disparity energy units to the local disparity networks and to the selection networks. The weights to the output units from the local disparity and selection pathways were fixed at 1.0. Once the model was trained, we obtained a scalar disparity estimate from the model by computing a nonlinear least squares Gaussian fit to the four output values. The mean of the Gaussian was our disparity estimate. When two objects were present

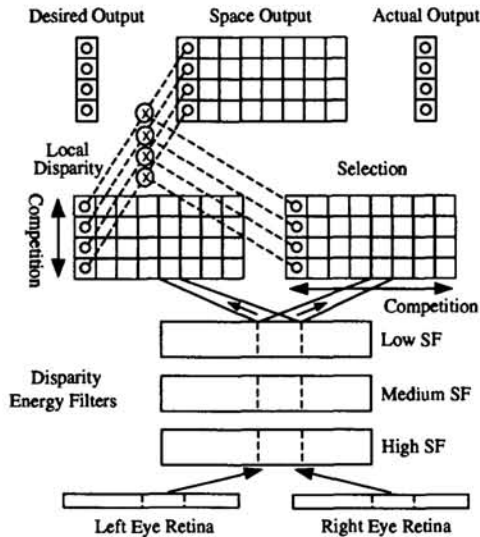

Figure 1: The mixture-of-experts architecture.

in the input, we fit the sum of two Gaussians to the four output values.

## 2.1   DISPARITY ENERGY FILTERS

The retinal layer in the model consisted of one-dimensional right eye and left eye images, each 82 pixels in length. These images were the input to disparity energy filters, as developed by Ohzawa, DeAngelis, and Freeman [7]. At the energy filter layer, there were 51 receptive field (RF) locations which received input from partially overlapping regions of the retinae. At each of these RF locations, there were 30 energy units corresponding to 10 phase differences at 3 spatial frequencies. These phase differences were proportional to disparity. An energy unit consisted of 4 energy filter pairs, each of which was a Gabor filter. The outputs of the disparity energy units were normalized at each RF location and within each spatial frequency using a soft-max nonlinearity.

## 2.2   LOCAL DISPARITY NETWORKS

In the local disparity pathway, there were 8 RF locations, and each received a weighted input from 9 disparity energy locations. Each RF location corresponded to a local disparity network and contained a pool of 4 disparity-tuned units. Neighboring locations received input from overlapping sets of disparity energy units. Weights were shared across all RF locations for each disparity. Soft-max competition occurred *within* each local disparity network (across disparity), and insured that only one disparity was strongly activated at each RF location.

## 2.3   SELECTION NETWORKS

Like the local disparity networks, the selection networks were organized into a grid of 8 RF locations with a pool of 4 disparity-tuned units at each location. These 4 units represented the local support for each of the different disparity hypotheses. It is more useful to think of the selection networks, however, as 4 separate layers each

of which responded to a specific disparity across all regions of the image. Like the local disparity pathway, neighboring RF locations received input from overlapping disparity energy units, and weights were shared across space for each disparity. In addition, the outputs of the selection network were normalized with the softmax operation. This competition, however, occurred separately for each of the 4 disparities in a global fashion *across space.*

## 3 RESULTS

Figure 2 shows the pattern of activations in the model when presented with a single object at a disparity of 2.1 pixels. The visual layout of the model in this figure is identical to the layout in Figure 1. The stimulus appears at bottom, with the 3 disparity energy filter banks directly above it. On the left above the disparity energy filters are the local disparity networks. The selection networks are on the right. The summed output (across space) appears in the upper right corner of the figure. Note that the selection network for a 2 pixel disparity (2nd row from the bottom in the selection pathway) is active for the spatial location at far left. The corresponding location is also highly active in the local disparity pathway, and this combination leads to strong activation for a 2 pixel disparity in the output of the model.

The mixture-of-experts model was optimized individually on a variety of different datasets and then tested on novel stimuli from the same datasets. The model's ability to discriminate among different disparities was quantified as the disparity threshold — the disparity difference at which one can correctly see a difference in depth 75% of the time. Disparity thresholds for the test stimuli were computed using signal-detection theory (Green & Swets [3]). Sample stimuli and their disparity thresholds are shown in Table 1. The model performed best on single object stimuli (top row). This disparity threshold (0.23 pixels) was substantially less than the input resolution of the model (1 pixel) and was thus exhibiting stereo hyperacuity. The model also performed well when there were multiple, occluding objects (2nd row). When both the stimulus and the background were generated from a uniform random distribution, the disparity threshold rose to 0.55 pixels. The model estimated disparity accurately in random-dot stereograms and real world images. Binary stereograms containing two transparent surfaces, however, were a challenging stimulus, and the threshold rose to 0.83 pixels. Part of the difficulty with this stimulus (containing two objects) was fitting the sum of 2 Gaussians to 4 data points.

We have compared our mixture-of-experts model (containing both a selection pathway and a local disparity pathway) with standard backpropagation and cross-correlation techniques (Gray et al [2]). The primary difference is that the backpropagation and cross-correlation models have no separate selection mechanism. In essence, one mechanism must compute both the segmentation and the disparity estimation. In our tests with the back-propagation model, we found that disparity thresholds for single object stimuli had risen by a factor of 3 (to 0.74 pixels) compared to the mixture-of-experts model. Disparity estimation of the cross-correlation model was similarly poor. Thresholds rose by a factor of 2 (compared to the mixture-of-experts model) for both single object stimuli and the noise stimuli (threshold = 0.46, 1.28 pixels, respectively).

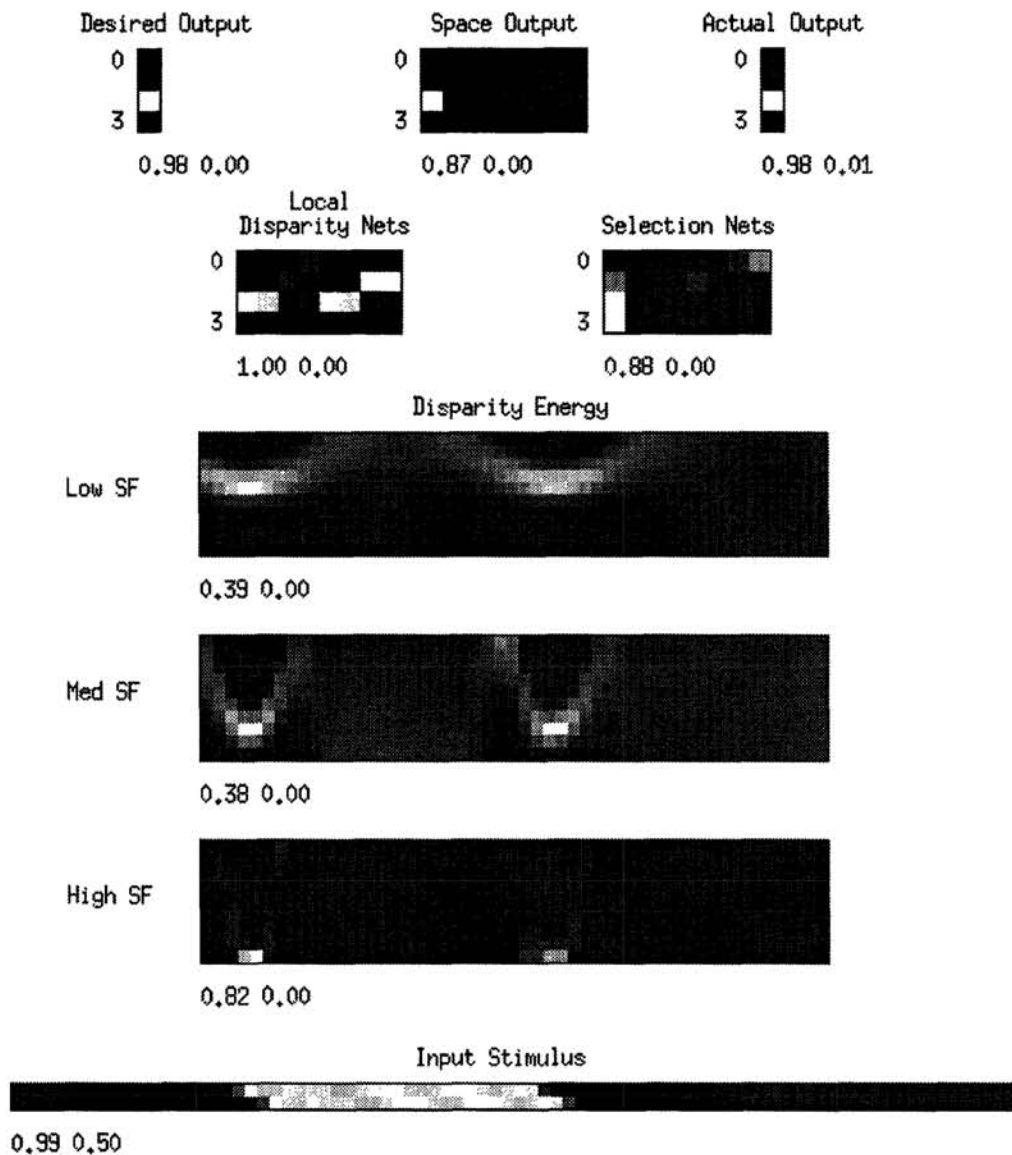

Figure 2: The activity in the mixture-of-experts model in response to an input stimulus containing a single object at a disparity of 2.10 pixels. At bottom is the input stimulus. The 3 regions in the middle represent the output of the disparity energy filters. Above the disparity energy output are the two pathways of the model. The local disparity networks appear to the left and the selection networks are to the right. Both the local disparity networks and the selection networks receive topographically organized input from the disparity energy filters. The selection and local disparity networks are displayed so that the top row represents a disparity of 0 pixels, the next row a 1 pixel disparity, then 2 and 3 pixel disparities in the remaining rows. At the top left part of the figure is the desired output for the given input stimulus. In the top middle is the output for each local region of space. On the top right is the actual output of the model collapsed across space. The numbers at the bottom left of each part of the network indicate the maximum and minimum activation values within that part. White indicates maximum activation level, black is minimum.

| Stimulus Type | Sample Stimulus | Threshold |
|:---:|:---:|:---:|
| Single |  | 0.23 |
| Double | 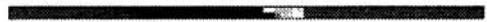 | 0.41 |
| Noise |  | 0.55 |
| Random-Dot |  | 0.36 |
| Transparent |  | 0.83 |
| Real |  | 0.30 |

Table 1: Sample stimuli for each of the datasets, and corresponding disparity thresholds (in pixels) for the mixture-of-experts model.

# 4 COMPARISON WITH NEUROPHYSIOLOGICAL DATA

To gain insight into the response properties of the selection units in our model, we mapped their activations as a function of space and disparity. Specifically, we

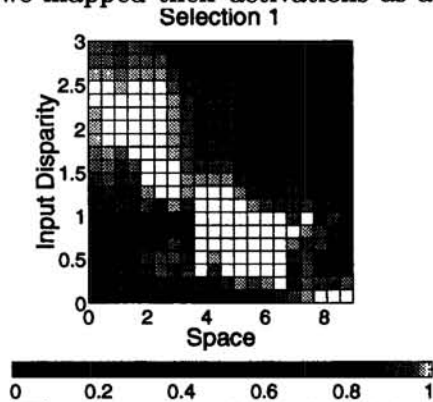

**Figure 3:** Selection unit activity

measured the activation of a unit as a single high-contrast edge was moved across the spatial extent of the receptive field. At each spatial location, we tested all possible disparities. An example of this mapping is shown in Figure 3. This selection unit is sensitive to changes in disparity as we move across space. We refer to this property as *disparity contrast*. In other words, the selection unit learned that a reliable indicator for a given disparity is a change in disparity across space. This type of detector can be behaviorally significant, because disparity contrast may play a role in signaling object boundaries. These selection units could thus provide valuable information in the construction of a 3-D model of the world. Recent neurophysiological studies by von der Heydt, Zhou, Friedman, and Poggio [8] is consistent with this interpretation. They found that neurons of awake, behaving monkeys in area V2 responded to edges of 4° by 4° random-dot stereograms. Because random-dot stereograms have no monocular form cues, these neurons must be responding to edges in depth. This sensitivity to edges in a depth map corresponds directly to the response profile of the selection units.

# 5 DISCUSSION

A major difficulty in estimating the disparities of objects in a visual scene in realistic circumstances (i.e., with clutter, transparency, occlusion, noise) is knowing which cues are most reliable and should be integrated, and which regions have ambiguous or unreliable information. Nowlan and Sejnowski [6] found that selection units learned to respond strongly to image regions that contained motion energy in several different directions. The role of those selection units is similar to layered analysis techniques for computing support maps in the motion domain (Darrell & Pentland [1]). The operation of the dual pathways in our model bears some similar-

ities to the pathways developed in the motion model of Nowlan and Sejnowski [6]. In the stereo domain, we have found that our selection units develop into edge detectors on a disparity map. They thus responded to regions rich in disparity information, analogous to the salient motion information captured in the motion [6] selection units.

We have also found that the model matches psychophysical data recorded by Westheimer and McKee [9] on the effects of spatial frequency filtering on disparity thresholds (Gray et al [2]). They found, in human psychophysical experiments, that disparity thresholds increased for any kind of spatial frequency filtering of line targets. In particular, disparity sensitivity was more adversely affected by high-pass filtering than by low-pass filtering.

In summary, we propose that the functional division into *local response* and *selection* represents a general principle for image interpretation and analysis that may be applicable to many different visual cues, and also to other sensory domains. In our approach to this problem, we utilized a multi-scale neurophysiologically-realistic implementation of binocular cells for the input, and then combined it with a neural network model to learn reliable cues for disparity estimation.

# References

[1] T. Darrell and A.P. Pentland. Cooperative robust estimation using layers of support. *IEEE Transactions on Pattern Analysis and Machine Intelligence*, 17(5):474–87, 1995.

[2] M.S. Gray, A. Pouget, R.S. Zemel, S.J. Nowlan, and T.J. Sejnowski. Reliable disparity estimation through selective integration. *INC Technical Report 9602, Institute for Neural Computation, University of California, San Diego*, 1996.

[3] D.M. Green and J.A. Swets. *Signal Detection Theory and Psychophysics*. John Wiley and Sons, New York, 1966.

[4] R.A. Jacobs, M.I. Jordan, S.J. Nowlan, and G.E. Hinton. Adaptive mixtures of local experts. *Neural Computation*, 3:79–87, 1991.

[5] R. Jain, R. Kasturi, and B.G. Schunck. *Machine Vision*. McGraw-Hill, New York, 1995.

[6] S.J. Nowlan and T.J. Sejnowski. Filter selection model for motion segmentation and velocity integration. *Journal of the Optical Society of America A*, 11(12):3177–3200, 1994.

[7] I. Ohzawa, G.C. DeAngelis, and R.D. Freeman. Stereoscopic depth discrimination in the visual cortex: Neurons ideally suited as disparity detectors. *Science*, 249:1037–1041, 1990.

[8] R. von der Heydt, H. Zhou, H. Friedman, and G.F. Poggio. Neurons of area V2 of visual cortex detect edges in random-dot stereograms. *Soc. Neurosci. Abs.*, 21:18, 1995.

[9] G. Westheimer and S.P. McKee. Stereoscopic acuity with defocus and spatially filtered retinal images. *Journal of the Optical Society of America*, 70:772–777, 1980.
